# Codec Avatar Studio: Paired Human Captures for Complete, Driveable, and Generalizable Avatars

Julieta Martinez[1], Emily Kim[2], Javier Romero[1],
Timur Bagautdinov[1], Shunsuke Saito[1], Shoou-I Yu[1], Stuart Anderson[1], Michael Zollhöfer[1],
Te-Li Wang[1], Shaojie Bai[1], Chenghui Li[1], Shih-En Wei[1], Rohan Joshi[1], Wyatt Borsos[1],
Tomas Simon[1], Jason Saragih[1], Paul Theodosis[1], Alexander Greene[1], Anjani Josyula[1],
Silvio Mano Maeta[1], Andrew I. Jewett[1], Simon Venshtain[1], Christopher Heilman[1],
Yueh-Tung Chen[1], Sidi Fu[1], Mohamed Ezzeldin A. Elshaer[1], Tingfang Du[1], Longhua Wu[1],
Shen-Chi Chen[1], Kai Kang[1], Michael Wu[1], Youssef Emad[1], Steven Longay[1], Ashley Brewer[1],
Hitesh Shah[1], James Booth[1], Taylor Koska[1], Kayla Haidle[1], Matt Andromalos[1],
Joanna Hsu[1], Thomas Dauer[1], Peter Selednik[1], Tim Godisart[1], Scott Ardisson[1],
Matthew Cipperly[1], Ben Humberston[1], Lon Farr[1], Bob Hansen[1], Peihong Guo[1],
Dave Braun[1], Steven Krenn[1], He Wen[1], Lucas Evans[1], Natalia Fadeeva[1], Matthew Stewart[1],
Gabriel Schwartz[1], Divam Gupta[1], Gyeongsik Moon[1], Kaiwen Guo[1], Yuan Dong[1],
Yichen Xu[1], Takaaki Shiratori[1], Fabian Prada[1], Bernardo R. Pires[1], Bo Peng[1],
Julia Buffalini[1], Autumn Trimble[1], Kevyn McPhail[1], Melissa Schoeller[1],
Yaser Sheikh[1]

[1]Codec Avatars Team, Meta Reality Labs Research
[2] Robotics Institute, Carnegie Mellon University

## Abstract

To create photorealistic avatars that users can embody, human modeling must be complete (encompass the full body), driveable (able to reproduce motion of the user from lightweight sensors), and generalizable (*i.e.*, easily adaptable to novel identities). Towards these goals, *paired* captures, that is, captures of the same subject obtained from systems of diverse quality and availability, are crucial. However, paired captures are rarely available to researchers outside of dedicated industrial labs: *Codec Avatar Studio* is our proposal to close this gap. Towards generalization and driveability, we introduce a dataset of 256 subjects captured in two modalities: high resolution multi-view scans of their heads, and video from the internal cameras of a headset. Towards completeness, we introduce a dataset of 4 subjects captured in eight modalities: high quality relightable multi-view captures of heads and hands, full body multi-view captures with minimal and regular clothes, and corresponding head, hands and body phone captures. Together with our data, we also provide code and pre-trained models for different state-of-the-art human generation models. Our datasets and code are available at `https://github.com/facebookresearch/ava-256` and `https://github.com/facebookresearch/goliath`.

## 1   Introduction

People are deeply social creatures. It is no coincidence that remote communication has frequently been one of the first applications of new technologies. Letters, telegrams, phone calls, instant messaging, and video conferencing are well-known examples of peer-to-peer communication technologies that have helped users stay in touch with the people they care about, and create new social connections.

Virtual reality (VR) and adjacent technologies promise to enable novel forms of remote social interaction. In particular, headsets with accurate localization let users navigate 3d environments naturally: by simply moving around. Coupled with virtual models of people (*i.e.*, photorealistic avatars) and a mechanism to drive them, it is possible to build systems where virtually-embodied users interact with each other in virtual 3d spaces. We can imagine that these environments will enable modes of communication more and more similar to in-person interactions once certain conditions are met: reduced latency, increased realism of the models, and more accurate control of those avatars.

We argue that, for photorealistic avatars to be deployed at scale, models of human appearance must fulfill certain conditions. They should be (1) complete, that is, not just be limited to faces and hands, but encompass full bodies. Models also need to be (2) driveable, that is, it must be possible to build mechanisms that track motion and appearance changes of the user with as little interference as possible, such that these changes can be transmitted compactly and reproduced on the other end. Finally, avatars must be (3) generalizable, as in it must be possible to build avatars for new users quickly, without access to expensive capture setups. To achieve these goals, *paired* captures, that is, captures of the same people using different devices are crucial. For example, to build or evaluate a mechanism that drives face avatars from a headset, it is necessary to collect data of the same people both in a high resolution scanner (to build a high quality avatar), and from the headset itself. Similarly, towards generalizable avatars, one may want to study the creation of high quality avatars from lower-quality but more widely available capture setups, such as phone scans. In that case, it is necessary to capture the same person under both a high resolution scanner, and from a phone scan. Unfortunately, paired captures are not usually available to researchers outside of specialized industrial labs, and we believe that this limitation has overall slowed down progress in avatar generation.

In this paper, our goal is to close this gap by providing the research community with a series of datasets of paired captures, as well as state-of-the-art implementations of avatar models. First, we introduce Ava-256, a dataset of 256 subjects, each captured in both a high resolution dome with dozens of views (meant for avatar creation), and while wearing a commercial headset with several infrared cameras (meant for driving). Second, we introduce Goliath-4, a dataset of 4 subjects, each captured under 8 modalities: A high resolution dome with relightable captures of heads and hands; two full body captures with regular and minimal clothing, and corresponding phone scans for heads, hands, and full body. Besides the raw data, we also provide several assets, such as 3d mesh tracking, semantic segmentation, and 3d keypoints. Moreover, we also provide code and pre-trained models for personalized relightable head [49], personalized relightable hand [23], and personalized full body [4] avatars, as well as multi-person head avatars [13], and an out-of-the-box driver from headset images [60]. Finally, the phone captures are compatible with systems for instant creation of head [13] and hand models [36]. Given its size and scope, we refer to our joint dataset and code release as *Codec Avatar Studio*. Our goal is for Codec Avatar Studio to become a toolkit that academics can use to bootstrap their engagement with the fundamental problems of photorealistic telepresence.

## 2 Datasets of Paired Human Captures

In this Section we introduce our head dataset, Ava-256, and our full-body dataset, Goliath-4.

**Subjects and license.** For all captures, participants were at least 18 years old at the time of the session, and provided their written informed consent for the capture and its distribution. Specifically, the consent forms contain our contact information, the purpose of the study, a description of the capture devices and capture procedures, and the estimated time the study will take. Additionally, the forms state that participation is voluntary, that participants may withdraw at any time and still get compensated for their time, that the data will be used for research purposes, and that Meta is the licenser and owner of the collected data, which may be distributed to third parties in the future. Note that participants keep a copy of the consent form after the study. Finally, the participants were compensated with USD $50 per hour, rounded to the next half hour. We estimate that participants were paid USD $14 000 in total. We release all our data and assets under the CC-by-NC 4.0 License.

### 2.1 Ava-256

Ava-256 is the first publicly available dataset of subjects collected in a high resolution dome as well as from a commercially available headset. Ava-256 was collected in a span of twenty-five months,

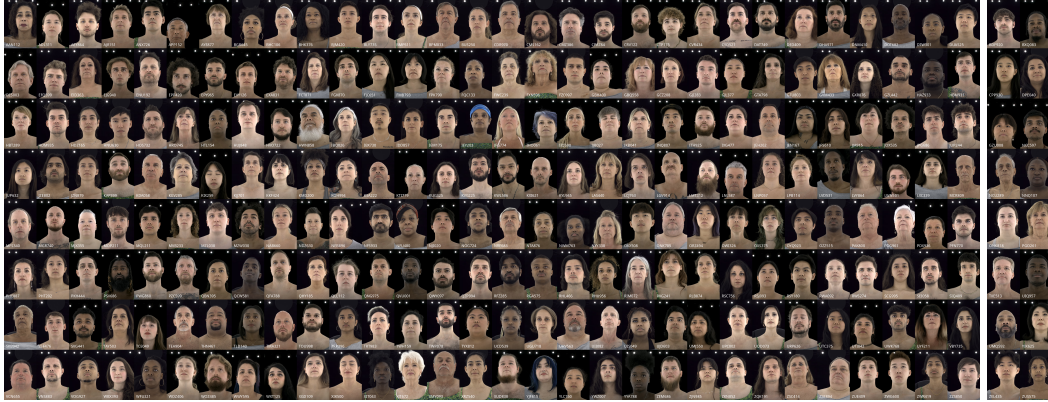

Figure 1: All 256 subjects in Ava-256. Notice the diversity in attributes such as hairstyle and hair length, makeup, jewellery, tattoos, facial hair, skin tone, and age. Left: proposed training split of 240 subjects for cross-id tasks. Right: proposed validation set of 16 subjects. Best viewed zoomed-in.

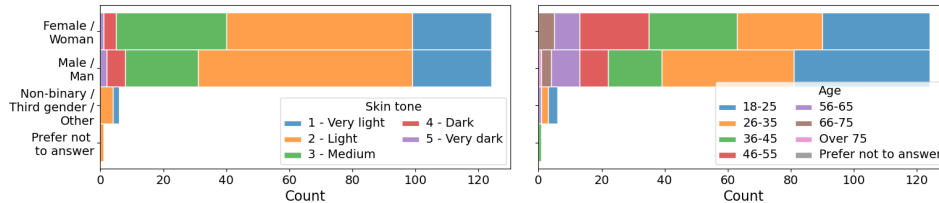

Figure 2: Self-reported skin tone and age distributions by gender in the Ava-256 dataset.

between August of 2021 and September of 2023 in Pittsburgh, PA. Figure 1 shows an image of all the subjects in Ava-256. Please refer to Figure 2 for a summary of self-reported age and skin tone diversity in our dataset.

**Dome captures.** Our capture setup is identical to the one described in [61, 13]. It consists of roughly 170 synchronized cameras placed in a dome with a 2.4m diameter. The cameras use a Sony IMX253 sensor with a pixel size of 3.45 µm, shutter speed of 2.222 ms and 35mm lenses. Each camera produces images of size $4\,096 \times 2\,668$ at 30 fps. A research assistant (RA) guides the subjects through a series of prompts to elicit a diverse set of expressions. These prompts include facial expressions, such as rolling their eyes, looking up and down, opening their mouths, and puffing their cheeks, longer-format conversations, cues to display specific emotions, and 20 to 30 phrases chosen to produce the span of phonemes commonly used in English. Each session lasts about half an hour. Some prompts may be repeated at the discretion of the RA, but only the best performance of each action, as deemed by the RA, is stored. This results in approximately $20\,000$ frames, or $\sim\!20\,000 \times 172 \approx 3.5\text{M}$ images per capture.

**Headset captures.** A similar script is recorded for each participant on a Quest Pro headset, which has 5 cameras: one camera per eye, two mouth cameras placed close to the cheekbones, and one camera looking at the glabella. The cameras capture infrared light, and their outputs are monochrome images of resolution $400 \times 400$ at 72 fps. For ground truth computation (*c.f.,* Section 3.2), we use an augmented headset with 10 cameras in total. The extra cameras provide additional view of each eye, an additional view of the glabella, and two additional views of the mouth from a less oblique angle. Figure 3 shows a frame captured from one of our headsets.

**Data distribution and compression.** One of the key issues with distributing large-scale image datasets is storage, which impacts both required memory and download times. For example, each of our dome captures produces over 10 TB of data when stored at full resolution and compressed losslessly. This configuration would produce over 3 PB in total for the dataset. Storing this amount of data is infeasible for most academic labs, and can be a major deterrent to use new datasets.

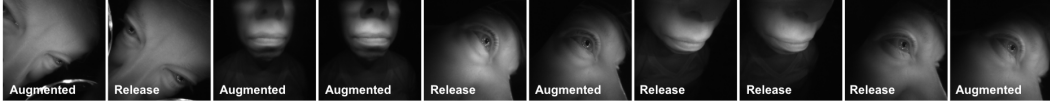

Figure 3: 10 frames captured by one of our augmented Quest Pro headsets. "Release" frames are included as part of ava-256, while "Augmented" ones are only used to aid in ground truth computation.

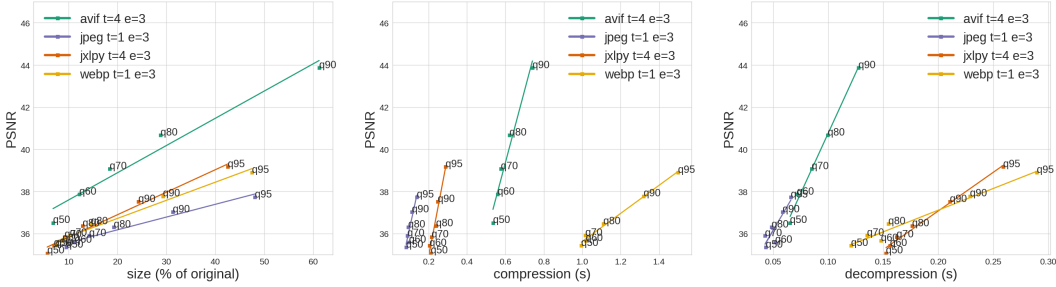

Figure 4: Compression comparison of WEBP, JPEG, JPEGXL and AVIF formats, the latter two using 4 threads. The points are measurements, and we fit a line to make it easier to spot the trend. For different levels of quality (qXX in the plot), AVIF is a clear winner for our data.

Following the trend from foundational models [11, 53, 55], we instead make 4 more accessible releases of 4, 8, 16, and 32 TB, each with different trade-offs of space against quality. We target a minimal size of 4TB since this is the current capacity of new hard and solid state drives under USD $50 and USD $200, respectively.[1] The higher quality versions may be used by researchers with more resources, and will become more accessible over time with decreasing storage costs.

For a given storage budget, our goal is to get the highest quality images into the hands of researchers. To determine the most suitable image format for our dataset, we selected 4 images from each capture, sampling cameras and frames uniformly at random, and evaluated the peak signal-to-noise ratio (PSNR), and compression and decompression times of WEBP, JPEG, JPEGXL, and AVIF formats – the latter two chosen for their user-friendly licenses (Fig. 4). Given its superior quality-to-space ratio, although higher yet acceptable decompression time, we have chosen to distribute our dataset in AVIF format. We are aware that this format is less commonly used in the machine learning community, and have thus dedicated engineering time to improve the most popular Pillow plugin for AVIF[2].

Compressing images is not enough, since our target release sizes would require compression parameters that would result in too low PSNR for avatar creation. To fit the data into reasonably sized packages we also downsampled the resolution of our images ($2\times$ and $4\times$), the number of cameras ($\sim 2\times$ down to 80), and the frames per second ($2\times$ and $4\times$, from 30 to 15 and 7.5). We chose cameras manually, making sure there is full $360°$ coverage of the head, and that there is always a frontal-facing camera. A summary of our Ava-256 releases can be found in Table 1.

Table 1: Details of our different size-friendly Ava-256 releases.

| | Dome (80 RGB cameras) | | | | Headset (5 IR cameras) | | | |
|---|---|---|---|---|---|---|---|---|
| Size | FPS | Image resolution | Frames per subject | AVIF quality | FPS | Image resolution | Frames per subject | AVIF quality |
| 4 TB | 7.5 | $1\,024 \times 667$ | $\sim 5\,000$ | 63 | 9 | $400 \times 400$ | $\sim 20\,000$ | 70 |
| 8 TB | 15.0 | $1\,024 \times 667$ | $\sim 10\,000$ | 63 | 18 | $400 \times 400$ | $\sim 40\,000$ | 70 |
| 16 TB | 7.5 | $2\,048 \times 1\,334$ | $\sim 5\,000$ | 70 | 9 | $400 \times 400$ | $\sim 20\,000$ | 70 |
| 32 TB | 15.0 | $2\,048 \times 1\,334$ | $\sim 10\,000$ | 70 | 18 | $400 \times 400$ | $\sim 40\,000$ | 70 |

## 2.2 Goliath-4

We complement the subject diversity and driveability of Ava-256 with Goliath-4, which focuses on extending the scope of the scans to the full body. Towards this goal, it is not enough to simply provide full body captures from a multiview scanner. First, since the resolution of multiview full-body captures in critical areas like head and hands is insufficient, we also provide captures of higher pixel density for those areas. Second, on top of the uniform illumination used in the body captures and Ava-256, Goliath-4 head and hand captures include interleaved shots with what we call "group lights": a variation of one-light-at-a-time (OLAT) in which we directionally illuminate the scene with a small group of adjacent lights, to increase the brightness and reduce aliasing compared to OLAT.

In full body captures, unlike head or hand scans, most of the skin is covered by clothes. This represents a challenge for some reconstruction methods, since body models [32] are often created from subjects in tight-fit clothing. In order to enable research on the relation between clothed and tight-fit clothing subjects, we additionally provide comprehensive minimally-clothed full body captures.

The data described so far can only be captured with expensive multiview scanning systems available in dedicated research labs, and is thus inaccessible to most researchers. To bridge this gap, we include a phone capture corresponding to each of the previously described ones—head, hands, clothed bodies and minimally clothed bodies. These types of captures already enable the creation of head and hand avatars for everyday users when put together with large scale high quality datasets [13, 36].

**Capture day**   Each subject is captured in three different rooms: the relightable scanner for heads and hands, a normal room for phone captures, and a full body scanner for bodies. The capture lasts for around 5 hours, including lunch breaks and transportation across scanners in different buildings.

**Data package**   The total recording time per participant is 135 minutes on average, which can be further broken down to 25, 25, 28, and 57-minutes for head, hand, normally clothed full body, and minimally clothed full body recordings respectively. The original sequences, compressed losslessly, take around 762 TB of storage per participant. Similar to Section 2.1, we compressed the original sequences with AVIF quality 63, lowered the image resolution 2x (from $4\,096 \times 2\,668$ to $2\,048 \times 1\,334$) and subsampled in time to produce effective training set sizes of roughly $10\,000$ frames.

**Head scans.**   The head scans were collected from the same system as Ava-256, following the same set of actions. The main difference is that the lighting is time-multiplexed instead of constant. Images are collected at 90 frames per second, while one out of every three frames is captured with all $\sim 400$ lights (SmartVision SX30) being on during 120 µs, and the other two out of three frames have either a "group" light or a random configuration of lights turned on. The group light [7] resembles OLAT [19] where one of the lights and its four nearest neighbors are turned on for 2.2 ms. The random configuration includes either 5 or 10 random lights turned on for 5 and 4 ms, respectively. We divide the captures into a train and test split. The test split includes a full segment of varied extreme face expressions, and thirty seconds of a conversation segment. The train split includes all remaining segments as well as four minutes of conversation. The conversation frames in the train split are not only disjoint from the test split, but also separated by a buffer of 30 seconds to avoid temporal leaking of information between the train and test splits. To enable a reasonable size footprint, the train split is subsampled at 10 frames per second.

**Hand scans.**   Hand scans are collected in the same system as the head scans, also in a time-multiplexed manner. In this case, half (instead of one third for the head captures) of the frames are fully-lit, while the other half use either group or random illuminations. We collected data for both right and left hand separately. The test split includes a segment with free hand motion, while the rest of the segments are included in the train split after being subsampled at 5 frames per second.

**Full-body scan.**   Collecting data for full-body motion requires a larger dome: This capture system has a diameter of 5.5m with $\sim 230$ cameras that acquire images at 30 frames per second under constant illumination. Subjects were captured twice, once in minimal clothing and once wearing their own garments, *i.e.*, normal clothing. The script followed in the normal clothing capture is a subset of the minimal one, where segments involving only head and hand motions have been removed. The test set contains a sequence where the subjects play charades, describing with their body motion

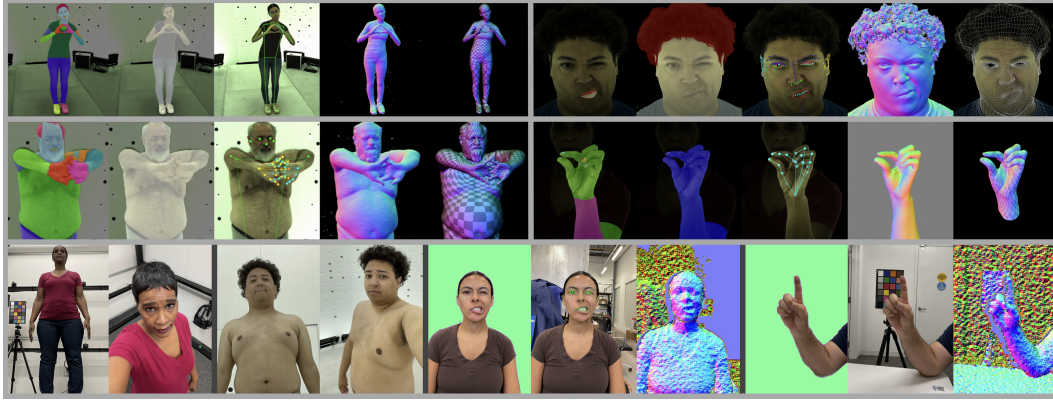

Figure 5: Goliath-4 Assets. The first two rows show full body clothed assets, minimal full body assets, head and hand assets. Each of the blocks show two segmentations, 3D keypoints, 3D reconstruction, and template registration. The last row shows phone assets: full body clothed and minimal (static picture and hand-held video), head and hands (segmentation, keypoints and depth).

specific concepts. The training set contains the rest of the sequences, and it is sampled at 10 frames per second for the clothed capture, and 5 frames per second for the minimal clothed capture.

**Phone scan.** Phone scans were taken in the full body scanner dome for minimal clothed full body, and in a separate room for the head, hand, and normally clothed full body sessions. The phone was placed on a tripod for all sequences in which it was not handheld. We collected RGB videos at $1\,080 \times 1\,920$ and RGB images at $2\,316 \times 3\,088$ resolution. The RGB videos contain also a depth video collected from the depth sensor of an iPhone 11, at resolution $360 \times 640$ and 30 frames per second.

**Assets.** For our multi-view data, we provide foreground-background and part segmentation, 3D keypoints, 3D reconstruction and registration (see Figure 5). Camera calibration is included, as well as light calibration for relightable sessions. Segmentation and 2D keypoints are provided for head and hand phone data, while no assets are provided for full body phone data in this release.

## 3 Code and Models

In this Section, we summarize the details of the multi-person face model as well as the the personalized head, hands, and body models, which are trained using our datasets. We release our code and models under the CC-by-NC 4.0 License.

### 3.1 Multi-identity face modelling

For high quality head avatars, we provide an implementation of the multi-person face model in Cao et al. [13]. Faces are represented as a collection of small semi-transparent cubes, or primitives, referred to as a Mixture of Volumetric Primitives (MVP) [31], which enable real-time rendering of dynamic volumetric content at high resolution.

The training setup aims to disentangle the identity and expressions of each subject, and to create a consistent expression space for multiple avatars. In particular, a 2D convolutional decoder progressively upsamples a latent *expression* code into two branches: (1) an appearance branch that outputs the RGB colour of each primitive, and (2) a geometry branch that outputs the vertices of a tracked mesh, as well as the 3d pose (relative to that mesh) and transparency of each primitive. The MVP representation enables end-to-end training of the model with a differentiable volumetric renderer.

To encourage a separation of identity and expression (see Fig. 6), the feature maps of the decoder are initialized by learned biases extracted from geometry and UV-texture maps of the neutral expression of the reconstructed subject. Similarly, a 2D convolutional variational encoder produces the expression codes from geometry and UV-texture maps subtracted from a canonical neutral frame. At test time, these codes can be replaced with ones produced by headset-mounted cameras, as described in 3.2.

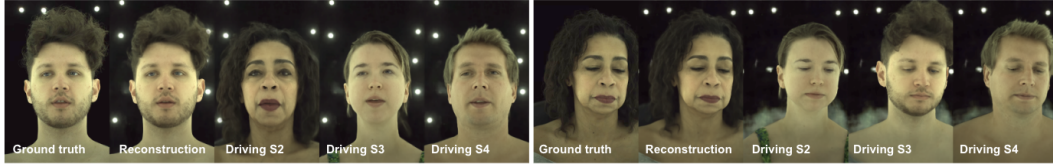

Figure 6: Two demonstrations of the consistent expression space produced by our multi-identity face model. Each panel shows the ground truth image, the reconstruction of the same subject, the same expression code rendered on three other subjects.

## 3.2 Generalizable face encoder

The generalizable face encoder maps headset IR images to the latent expression space learned by the multi-identity face model. A key challenge here is obtaining a reliable pseudo ground truth mapping from headset images to face deformations, after which the problem can be posed as supervised learning. We generate this pseudo ground truth using the method of Wei et al. [60], based on cycle consistency [70] between renders of personalized avatar models [30] and images from a headset. By jointly solving for the expression code and the relative head pose of each frame, as well as for the domain-transfer between the dome and headset images, we obtain person-specific latent expression codes for all the frames of a headset capture.

To train a multi-person encoder, we follow a simplified version of the method due to Bai et al. [5], where we relate the person-specific codes to the multi-person expression codes via pseudo-frontal renderings of the avatars. Specifically,

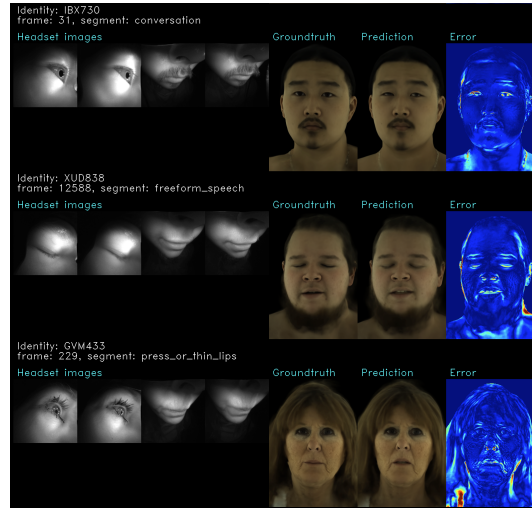

Figure 7: Generalizable encoder driving 3 subjects from headset images.

for each training sample, we render the person-specific decoder with a randomly sampled frontal camera. We then train an encoder that maps headset images to the multi-person latent expression space. This encoder is supervised by decoding the latent expressions using the universal face decoder from Section 3.1, and comparing it with the rendered person-specific avatar. The model is trained for all frames of the headset captures, across all training subjects, to produce a single generalizable encoder that can be applied to headset inputs from unseen subjects.

Our release includes person-specific codes for all Ava-256 headset captures, as well as the frontal renderings of the person-specific decoders to facilitate explorations of new generalizable encoders. We also provide code and assets necessary to train a generalizable encoder. Please refer to Figure 7 for a visualization of the provided driver.

## 3.3 Personalized relightable Gaussian heads

The relightable head avatars are based on Relightable Gaussian Codec Avatars (RGCA) [49], which achieve state-of-the-art performance on avatar modeling and relighting. The geometric representation is based on 3D Gaussians, which can be rendered with EWA splatting [73, 25] in real-time. This representation is particularly suitable to model thin structures such as skin details and hair strands. To support avatar relighting, RGCA models global light transport as learnable diffuse spherical harmonics and specular spherical Gaussians, which enables relighting from arbitrary light sources in real-time despite being trained on discrete point lights. Besides from controlling the environment light, the expression of the avatars can be modified through a latent vector, as with regular avatars. See Figure 8b for a qualitative sample of this model.

### 3.4 Personalized relightable hands with MVP

We base the relightable hand avatars on RelightableHands [23]. This work generalizes across different poses and light conditions in the presence of hand articulation by incorporating explicit shadow features computed by Deep Shadow Maps [29]. The hand geometry is represented by an articulated MVP [46] to support both articulation and volumetric modeling without compromising the efficiency of rendering. Hand pose is represented by joint angles and can be easily transferred across subjects. See Figure 8a for a qualitative sample of this model.

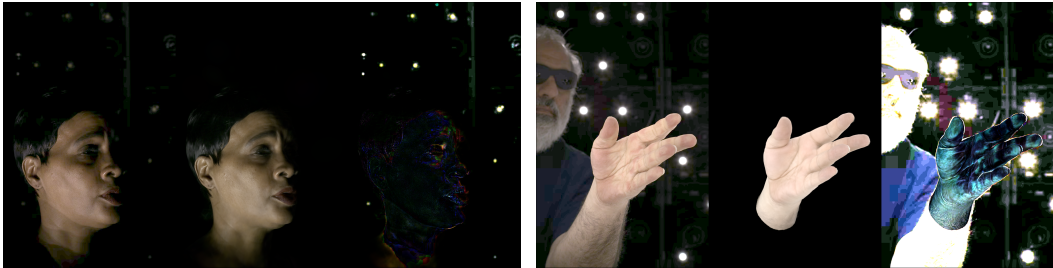

(a) Relightable face model of subject QZX685    (b) Personalized hand model of subject AXE977

Figure 8: Qualitative results of hand and face models based on mixtures of volumetric primitives (left) and Gaussian splatting (right). Ground truth, model prediction, and difference image.

### 3.5 Personalized full-body decoders

Full-body person-specific avatars are based on driving-signal-aware codec avatars [4]. This work uses a mesh-based neural representation. A body mesh and a view-dependent texture are decoded with a pose and latent code conditioned convolutional neural network, which are then rendered with an efficient differentiable renderer [43]. Generalization is achieved through a collection of disentangling techniques – localizing pose-dependent deformations, separating non-pose-dependent deformations into a disentengled latent space, and using an explicit physics-inspired prior for shadow modelling. See to Figure 9 for qualitative examples of these models.

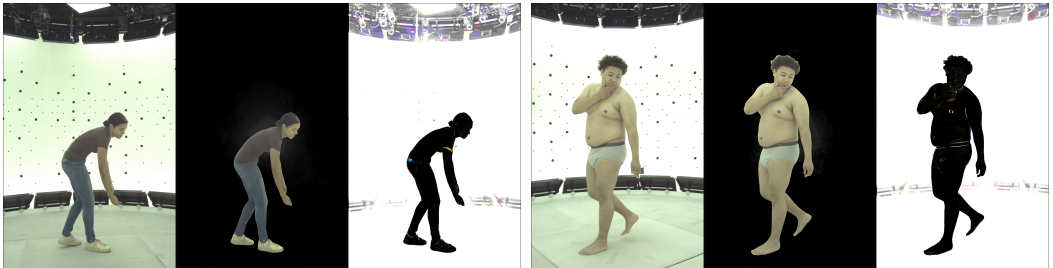

(a) Fully clothed body model of subject QVC422    (b) Minimally clothed body model of subject XKT970

Figure 9: Qualitative results of full-body models based on mesh and neural texture. Ground truth, model prediction, and difference image.

## 4 Limitations and Potential Negative Impact

Codec Avatar Studio opens numerous exciting research opportunities that were previously unavailable to academic researchers, but some challenges remain unsolved. For example, while ava-256 supports the creation of multi-person driveable decoders and Goliath-4 supports the creation of relightable head decoders, it will likely be hard to create multi-identity relightable head decoders with our release. Our datasets also fail to capture the long tail of human expressivity, such as the ability to convey sweat, tiredness, and blood flow. Goliath-4 allows experimentation with converting low-quality data (e.g., from mobile or full-body captures) into higher-quality outputs (such as multi-view head and hands). However, its sample size is limited to four subjects, restricting the scope of this research.

**Potential negative impact.**   Our releases share potential negative impact with other research related to the modelling of human appearance. In particular, more accurate controllable models of human appearance could be misused for impersonation or to faciliate the spread of disinformation. Potential mitigations may include authentication protocols for verifying the origin of information, avatar watermarking, and research into computer interfaces that clearly communicate whether media has been produced using photorealistic avatars.

# 5   Related Work

In this Section, we review the various types of human datasets for avatar generation. The discussion is divided into three parts: head, body, and hands datasets.

## 5.1   Datasets for Head Avatar Generation

Table 2: Summary of multi-view face datasets. Smaller scale dataset (fewer than 60 ids or 20 views) as shown first, then large-scale datasets, then ours.

| Dataset name | IDs | Views | Sentences + expressions | Images | Resolution | Size | Headset captures |
|---|---|---|---|---|---|---|---|
| D3DFACS[6] [18] | 10 | 6 | 19-97 | - | $1\,280 \times 1\,024$ | - | ✗ |
| CMU Multi-PIE [20] | 337 | 15 | 6 | 750K | $3\,072 \times 2\,048$ | 0.3 TB | ✗ |
| 4DFAB[6] [16] | 180 | 7 | 15 | - | $1\,600 \times 1\,200$ | - | ✗ |
| MEAD [58] | 48 | 7 | (8) | - | $1\,920 \times 1\,080$ | 0.9 TB | ✗ |
| Interdigital Light-Field [48] | 5 | 16 | - | - | $2\,048 \times 1\,088$ | 0.1 TB | ✗ |
| Multiface [61] | 13 | 40/150 | 168 | 15M | $2\,048 \times 1\,334$ | 65.0 TB | ✗ |
| NerSemble [26] | 222 | 16 | 25 | 31.7M | $3\,208 \times 2\,200$ | 1.0 TB | ✗ |
| HUMBI Face [67] | 617[1] | 68 | 20 | 17.3M | $200 \times 150$[5] | 1.3 TB | ✗ |
| Facescape [63] | 359[2] | 68 | 20 | 400K | $4\,344 \times 2\,896$ | 0.9 TB | ✗ |
| i3DMM[4] [65] | 64 | 137 | 10 | - | - | - | ✗ |
| RenderMe-360 [41] | 500[3] | 60 | 37/54 | 243M | $2\,448 \times 2\,048$ | 5.8 TB | ✗ |
| **Ava-256 Dome Captures (Ours)** | 256 | 80 | 35 | 217 M | $2\,048 \times 1\,334$ | 32.0 TB | ✓ |

[1] We only found the data of 403 subjects available online. We have contacted the authors to clarify this discrepancy.
[2] Out of 847 total subjects captured, only 359 subjects are available with multi-view image data.
[3] 500 captures are expected to be released later in 2024, but only 21 subjects were available online at time of submission. 5.8 TB is the size of this smaller subset containing about 4.2 % of the data, so we expect the final release size to be well over 100 TB.
[4] We requested access to the dataset on May 20th but have not heard back at the time of submission, and are thus unable to obtain more data about this dataset.
[5] The head images are cropped from the full-body $1\,920 \times 1\,080$ image, and are thus of low resolution.
[6] Dataset was not available on the website.

The growing interest in developing high-quality human head and face models has led to the collection of large-scale, multi-view head and face capture datasets. One notable example is HUMBI-Face [67], which captured facial expressions using 68 HD cameras from 772 subjects at a resolution of $1\,920 \times 1\,080$, significantly increasing the scale of human datasets. However, HUMBI-Face, while expanding the number of views and subjects, only captured 20 expressions and excluded conversations or speech. Similarly, Facescape [63] recorded avatar data from 847 subjects with 20 expressions at a resolution of $4\,344 \times 2\,896$, but released multi-view image data for only 359 subjects. On the other hand, i3DMM [65] involved fewer subjects and expressions, with 64 subjects performing 10 expressions, but featured a higher number of views with 137 camera angles. RenderMe-360 [41] is the more recent large-scale subject for face modelling; it aimed to enhance the diversity of subject appearance by varying hairstyles with wigs and incorporating various makeup styles among 500 subjects. The authors captured 12 expressions plus 25 to 42 sentences, using 60 camera views recorded at a resolution of $2\,448 \times 2\,048$. We summarize these datasets, plus others which are smaller in scale (with fewer than 60 IDs or fewer than 10 views) in Table 2.

Our Ava-256 dataset aims to provide comparable variability in the number of subjects, camera views, and expressions, while also including headset data that no previous dataset offers. Additionally, we provide four distribution-friendly release sizes to facilitate accessibility and long-term maintenance.

## 5.2 Datasets for Body Avatar Generation

The interest in human anthropometrics drove the creation of the first large full body datasets. One of the first large data collection efforts was CAESAR [47], composed by three colored laser scans for thousands of subjects. Private companies like Humanetics have continued similar studies devoted to anthromopetrics, collecting thousands of full body scans across Europe and North-America [22]. In academia, one of the first uses of 3D full body datasets was the creation of statistical body models. Anguelov et al. [2] created SCAPE, a single subject deformable model from 71 registered meshes of different poses. Allen et al. [1] created a multi-subject geometry model by combining data from 44 different subjects with 80 more scans of 5 subjects in different poses. Later models increased the amounts of data used for statistical full body models to hundreds [21], thousands [32], more than ten thousand [39], tens of thousands [62], and a million [40] samples. Other common use of geometry-only datasets have been modeling clothes [69, 33]. The increasing size of these datasets has resulted in improved accuracy and generalizability of the body models. However, the data typically lacks high quality appearance information, making these models inadequate for image generation.

Laser scans in the early 2 000s were replaced by multi-view photogrammetry scans, which capture high quality imagery as well as geometry (based on either active or passive stereo). One of the first uses of images in these captures was to improve the registration to a deformable template [8, 9]. However, thanks to deep neural networks, full body avatars that model not only geometry but also appearance became more common. X-Avatar [51] created an animatable implicit human avatar including geometry and appearance based on the X-Humans dataset, which includes more than 35 500 scans. A large number of systems and datasets have been devoted to model, replace or remove garments. Multiple synthetic datasets have been proposed to learn garment geometry and dynamics [6, 72, 50]. To estimate body shape under clothing, [64, 69] collected each six subjects performing multiple motions both in minimal clothing as well as other garments. Our proposal of collecting minimal and casual data is similar to theirs, but using more cameras of higher resolution and complementing with other modalities (i.e. phone, relightable heads and hands). The THuman3.0 dataset [52] improved scan quality, and garment variety over its previous versions [66, 52] to reconstruct and model garments from scans. Sizer [54] shares the same goal, for which it collected 2 000 scans from 100 subjects. 4DHumanOutfit [3] focused on large motions of more than thousand sequences thanks to their ample color scan space. Recently, 4D-Dress [59] released 78 000 scans focused on garment variety, including 64 different outfits and corresponding semantic vertex labels.

The closest effort to Goliath-4 is Humman [12], which collects a multimodal dataset including mobile phone data, multiview RGBD and handheld scans for 1 000 subjects resulting in more than sixty million frames in total. While their system focuses on subject variety, Goliath-4 provides higher quality data for each of our four subjects, including high quality geometry and texture for each full-body frame, as well as high quality relightable head and hand sessions.

## 5.3 Datasets for Hand Avatar Generation

The popularity of applications like 3D hand estimation from images have fostered the appearance of a large number of datasets devoted to hands. However, most of these datasets are composed by real images annotated with either joint positions [68, 57, 38, 27] or 3D model parameters [42, 71, 14, 10, 56], and dedicated almost exclusively to model the geometry of the hand. HTML [45] used 51 handheld scans to create a PCA space of the right hand texture. Handy [44] increased the variety and quality of their models by collecting more than 1 200 scans with a 3dMD multiview scanner which included high quality textures. The release of Interhand2.6M [34] enabled the recent creation of multiple high quality appearance models [37, 17, 15, 24, 28]. However, this dataset has two drawbacks: its illumination is fixed (which is improved with [35], but for synthetic data), and it does not include the rest of the body. We fix those two shortcomings in Goliath-4, although Interhand2.6M is still valuable given it contains hand interaction and larger subject variety than our proposed dataset.

## Footnotes

[1]Lower end of quotes from `https://diskprices.com/`, retrieved June 1, 2024.

[2]`https://github.com/fdintino/pillow-avif-plugin/`.

## References

[1] B. Allen, B. Curless, Z. Popović, and A. Hertzmann. Learning a correlated model of identity and pose-dependent body shape variation for real-time synthesis. In *Proceedings of the ACM SIGGRAPH symposium*, pages 147–156. Citeseer, 2006.

[2] D. Anguelov, P. Srinivasan, D. Koller, S. Thrun, J. Rodgers, and J. Davis. SCAPE: shape completion and animation of people. In *Proceedings of the ACM SIGGRAPH*, pages 408–416. ACM, 2005.

[3] M. Armando, L. Boissieux, E. Boyer, J.-S. Franco, M. Humenberger, C. Legras, V. Leroy, M. Marsot, J. Pansiot, S. Pujades, et al. 4DHumanOutfit: A multi-subject 4D dataset of human motion sequences in varying outfits exhibiting large displacements. *Computer Vision and Image Understanding*, 237:103836, 2023.

[4] T. Bagautdinov, C. Wu, T. Simon, F. Prada, T. Shiratori, S.-E. Wei, W. Xu, Y. Sheikh, and J. Saragih. Driving-signal aware full-body avatars. *ACM Transactions on Graphics*, 40(4), jul 2021. ISSN 0730-0301. doi: 10.1145/3450626.3459850. URL https://doi.org/10.1145/3450626.3459850.

[5] S. Bai, T.-L. Wang, C. Li, A. Venkatesh, T. Simon, C. Cao, G. Schwartz, R. Wrench, J. Saragih, Y. Sheikh, and S.-E. Wei. Universal facial encoding of codec avatars from vr headsets. *ACM Transactions on Graphics (ToG)*, 43:1–22, 2024.

[6] H. Bertiche, M. Madadi, and S. Escalera. CLOTH3D: clothed 3D humans. In *Proceedings of the European Conference on Computer Vision*, pages 344–359. Springer, 2020.

[7] S. Bi, S. Lombardi, S. Saito, T. Simon, S.-E. Wei, K. Mcphail, R. Ramamoorthi, Y. Sheikh, and J. Saragih. Deep relightable appearance models for animatable faces. *ACM Transactions on Graphics*, 40(4):1–15, 2021.

[8] F. Bogo, J. Romero, M. Loper, and M. J. Black. FAUST: Dataset and evaluation for 3D mesh registration. In *Proceedings of the IEEE/CVF Conference on Computer Vision and Pattern Recognition*, pages 3794–3801, 2014.

[9] F. Bogo, J. Romero, G. Pons-Moll, and M. J. Black. Dynamic FAUST: Registering human bodies in motion. In *Proceedings of the IEEE/CVF Conference on Computer Vision and Pattern Recognition*, July 2017.

[10] S. Brahmbhatt, C. Tang, C. D. Twigg, C. C. Kemp, and J. Hays. ContactPose: A dataset of grasps with object contact and hand pose. In *Proceedings of the European Conference on Computer Vision*, pages 361–378. Springer, 2020.

[11] T. Brown, B. Mann, N. Ryder, M. Subbiah, J. D. Kaplan, P. Dhariwal, A. Neelakantan, P. Shyam, G. Sastry, A. Askell, et al. Language models are few-shot learners. *Advances in neural information processing systems*, 33:1877–1901, 2020.

[12] Z. Cai, D. Ren, A. Zeng, Z. Lin, T. Yu, W. Wang, X. Fan, Y. Gao, Y. Yu, L. Pan, et al. Humman: Multi-modal 4D human dataset for versatile sensing and modeling. In *Proceedings of the European Conference on Computer Vision*, pages 557–577. Springer, 2022.

[13] C. Cao, T. Simon, J. K. Kim, G. Schwartz, M. Zollhoefer, S.-S. Saito, S. Lombardi, S.-E. Wei, D. Belko, S.-I. Yu, Y. Sheikh, and J. Saragih. Authentic volumetric avatars from a phone scan. *ACM Transactions on Graphics*, 41(4), jul 2022. ISSN 0730-0301. doi: 10.1145/3528223.3530143. URL https://doi.org/10.1145/3528223.3530143.

[14] Y.-W. Chao, W. Yang, Y. Xiang, P. Molchanov, A. Handa, J. Tremblay, Y. S. Narang, K. Van Wyk, U. Iqbal, S. Birchfield, et al. DexYCB: A benchmark for capturing hand grasping of objects. In *Proceedings of the IEEE/CVF Conference on Computer Vision and Pattern Recognition*, pages 9044–9053, 2021.

[15] X. Chen, B. Wang, and H.-Y. Shum. Hand Avatar: Free-pose hand animation and rendering from monocular video. In *Proceedings of the IEEE/CVF Conference on Computer Vision and Pattern Recognition*, pages 8683–8693, 2023.

[16] S. Cheng, I. Kotsia, M. Pantic, and S. Zafeiriou. 4DFAB: A large scale 4D database for facial expression analysis and biometric applications. In *Proceedings of the IEEE/CVF Conference on Computer Vision and Pattern Recognition*, June 2018.

[17] E. Corona, T. Hodan, M. Vo, F. Moreno-Noguer, C. Sweeney, R. Newcombe, and L. Ma. Lisa: Learning implicit shape and appearance of hands. In *Proceedings of the IEEE/CVF Conference on Computer Vision and Pattern Recognition*, pages 20533–20543, 2022.

[18] D. Cosker, E. Krumhuber, and A. Hilton. A FACS valid 3D dynamic action unit database with applications to 3D dynamic morphable facial modeling. In *Proceedings of the IEEE/CVF International Conference on Computer Vision*, pages 2296–2303, 11 2011. doi: 10.1109/ICCV. 2011.6126510.

[19] P. Debevec, T. Hawkins, C. Tchou, H.-P. Duiker, W. Sarokin, and M. Sagar. Acquiring the reflectance field of a human face. In *Proceedings of the 27th Annual Conference on Computer Graphics and Interactive Techniques*, pages 145–156, 2000.

[20] R. Gross, I. Matthews, J. Cohn, T. Kanade, and S. Baker. Multi-PIE. In *Proceedings of the IEEE International Conference on Automatic Face & Gesture Recognition*, pages 1–8, 2008. doi: 10.1109/AFGR.2008.4813399.

[21] N. Hasler, C. Stoll, M. Sunkel, B. Rosenhahn, and H.-P. Seidel. A statistical model of human pose and body shape. In *Computer Graphics Forum*, volume 28, pages 337–346. Wiley Online Library, 2009.

[22] Humanetics. SizeWorld. https://www.avalution.net/en/fashion/-completed/index.html, 2024. Accessed: 2024-06-01.

[23] S. Iwase, S. Saito, T. Simon, S. Lombardi, T. Bagautdinov, R. Joshi, F. Prada, T. Shiratori, Y. Sheikh, and J. Saragih. RelightableHands: Efficient neural relighting of articulated hand models. In *Proceedings of the IEEE/CVF Conference on Computer Vision and Pattern Recognition*, pages 16663–16673, 2023.

[24] K. Karunratanakul, S. Prokudin, O. Hilliges, and S. Tang. HARP: Personalized hand reconstruction from a monocular rgb video. In *Proceedings of the IEEE/CVF Conference on Computer Vision and Pattern Recognition*, pages 12802–12813, 2023.

[25] B. Kerbl, G. Kopanas, T. Leimkühler, and G. Drettakis. 3D gaussian splatting for real-time radiance field rendering. *ACM Transactions on Graphics*, 42(4):1–14, 2023.

[26] T. Kirschstein, S. Qian, S. Giebenhain, T. Walter, and M. Nießner. NeRSemble: Multi-view radiance field reconstruction of human heads. *ACM Transactions on Graphics*, 42(4), jul 2023. ISSN 0730-0301. doi: 10.1145/3592455. URL https://doi.org/10.1145/3592455.

[27] T. Kwon, B. Tekin, J. Stühmer, F. Bogo, and M. Pollefeys. H2O: Two hands manipulating objects for first person interaction recognition. In *Proceedings of the IEEE/CVF International Conference on Computer Vision*, pages 10138–10148, 2021.

[28] D. Lin, Y. Zhang, M. Li, Y. Liu, W. Jing, Q. Yan, Q. Wang, and H. Zhang. 4DHands: Reconstructing interactive hands in 4D with transformers. *arXiv preprint arXiv:2405.20330*, 2024.

[29] T. Lokovic and E. Veach. Deep shadow maps. In *Seminal Graphics Papers: Pushing the Boundaries, Volume 2*, pages 311–318. ACM, 2023.

[30] S. Lombardi, J. Saragih, T. Simon, and Y. Sheikh. Deep appearance models for face rendering. *ACM Transactions on Graphics*, 37(4):68:1–68:13, July 2018. ISSN 0730-0301.

[31] S. Lombardi, T. Simon, G. Schwartz, M. Zollhoefer, Y. Sheikh, and J. Saragih. Mixture of volumetric primitives for efficient neural rendering. *ACM Transactions on Graphics*, 40(4), jul 2021. ISSN 0730-0301. doi: 10.1145/3450626.3459863. URL https://doi.org/10.1145/3450626.3459863.

[32] M. Loper, N. Mahmood, J. Romero, G. Pons-Moll, and M. J. Black. SMPL: a skinned multi-person linear model. *ACM Transactions on Graphics*, 34(6), oct 2015. ISSN 0730-0301. doi: 10.1145/2816795.2818013. URL https://doi.org/10.1145/2816795.2818013.

[33] Q. Ma, J. Yang, A. Ranjan, S. Pujades, G. Pons-Moll, S. Tang, and M. J. Black. Learning to Dress 3D People in Generative Clothing. In *Proceedings of the IEEE/CVF Conference on Computer Vision and Pattern Recognition*, 2020.

[34] G. Moon, S.-I. Yu, H. Wen, T. Shiratori, and K. M. Lee. InterHand2.6M: A dataset and baseline for 3D interacting hand pose estimation from a single rgb image. In *Proceedings of the European Conference on Computer Vision*, pages 548–564. Springer, 2020.

[35] G. Moon, S. Saito, W. Xu, R. Joshi, J. Buffalini, H. Bellan, N. Rosen, J. Richardson, M. Mize, P. De Bree, et al. A dataset of relighted 3D interacting hands. *Advances in Neural Information Processing Systems*, 36, 2024.

[36] G. Moon, W. Xu, R. Joshi, C. Wu, and T. Shiratori. Authentic hand avatar from a phone scan via universal hand model. *arXiv preprint arXiv:2405.07933*, 2024.

[37] A. Mundra, J. Wang, M. Habermann, C. Theobalt, M. Elgharib, et al. LiveHand: Real-time and photorealistic neural hand rendering. In *Proceedings of the IEEE/CVF International Conference on Computer Vision*, pages 18035–18045, 2023.

[38] T. Ohkawa, K. He, F. Sener, T. Hodan, L. Tran, and C. Keskin. AssemblyHands: Towards egocentric activity understanding via 3D hand pose estimation. In *Proceedings of the IEEE/CVF Conference on Computer Vision and Pattern Recognition*, pages 12999–13008, 2023.

[39] A. A. Osman, T. Bolkart, and M. J. Black. STAR: Sparse trained articulated human body regressor. In *Proceedings of the European Conference on Computer Vision*, pages 598–613. Springer, 2020.

[40] A. A. Osman, T. Bolkart, D. Tzionas, and M. J. Black. SUPR: A sparse unified part-based human representation. In *Proceedings of the European Conference on Computer Vision*, pages 568–585. Springer, 2022.

[41] D. Pan, L. Zhuo, J. Piao, H. Luo, W. Cheng, Y. Wang, S. Fan, S. Liu, L. Yang, B. Dai, Z. Liu, C. C. Loy, C. Qian, W. Wu, D. Lin, and K.-Y. Lin. RenderMe-360: Large digital asset library and benchmark towards high-fidelity head avatars. In *Thirty-seventh Conference on Neural Information Processing Systems Datasets and Benchmarks Track*, 2023.

[42] G. Pavlakos, D. Shan, I. Radosavovic, A. Kanazawa, D. Fouhey, and J. Malik. Reconstructing hands in 3D with transformers. *arXiv preprint arXiv:2312.05251*, 2023.

[43] S. Pidhorskyi, T. Simon, G. Schwartz, H. Wen, Y. Sheikh, and J. Saragih. Rasterized edge gradients: Handling discontinuities differentiably, 2024.

[44] R. A. Potamias, S. Ploumpis, S. Moschoglou, V. Triantafyllou, and S. Zafeiriou. Handy: Towards a high fidelity 3D hand shape and appearance model. In *Proceedings of the IEEE/CVF Conference on Computer Vision and Pattern Recognition*, pages 4670–4680, 2023.

[45] N. Qian, J. Wang, F. Mueller, F. Bernard, V. Golyanik, and C. Theobalt. HTML: A parametric hand texture model for 3D hand reconstruction and personalization. In *Proceedings of the European Conference on Computer Vision*, pages 54–71. Springer, 2020.

[46] E. Remelli, T. Bagautdinov, S. Saito, C. Wu, T. Simon, S.-E. Wei, K. Guo, Z. Cao, F. Prada, J. Saragih, et al. Drivable volumetric avatars using texel-aligned features. In *Proceedings of the ACM SIGGRAPH*, pages 1–9, 2022.

[47] K. M. Robinette, S. Blackwell, H. Daanen, M. Boehmer, S. Fleming, T. Brill, D. Hoeferlin, and D. Burnsides. Civilian american and european surface anthropometry resource (CAESAR), final report, volume I: Summary. *Sytronics Inc Dayton Oh*, page 3, 2002.

[48] N. Sabater, G. Boisson, B. Vandame, P. Kerbiriou, F. Babon, M. Hog, T. Langlois, R. Gendrot, O. Bureller, A. Schubert, and V. Allie. Dataset and pipeline for multi-view light-field video. In *Proceedings of the IEEE/CVF Conference on Computer Vision and Pattern Recognition Workshops*, 2017.

[49] S. Saito, G. Schwartz, T. Simon, J. Li, and G. Nam. Relightable Gaussian Codec Avatars. In *Proceedings of the IEEE/CVF Conference on Computer Vision and Pattern Recognition*, 2024.

[50] Y. Shao, C. C. Loy, and B. Dai. Towards multi-layered 3D garments animation. In *Proceedings of the IEEE/CVF International Conference on Computer Vision*, pages 14361–14370, 2023.

[51] K. Shen, C. Guo, M. Kaufmann, J. Zarate, J. Valentin, J. Song, and O. Hilliges. X-Avatar: Expressive human avatars. *Proceedings of the IEEE/CVF Conference on Computer Vision and Pattern Recognition*, 2023.

[52] Z. Su, T. Yu, Y. Wang, and Y. Liu. Deepcloth: Neural garment representation for shape and style editing. *IEEE Transactions on Pattern Analysis and Machine Intelligence*, 45(2):1581–1593, 2022.

[53] G. Team, R. Anil, S. Borgeaud, Y. Wu, J.-B. Alayrac, J. Yu, R. Soricut, J. Schalkwyk, A. M. Dai, A. Hauth, et al. Gemini: a family of highly capable multimodal models. *arXiv preprint arXiv:2312.11805*, 2023.

[54] G. Tiwari, B. L. Bhatnagar, T. Tung, and G. Pons-Moll. Sizer: A dataset and model for parsing 3D clothing and learning size sensitive 3D clothing. In *Proceedings of the European Conference on Computer Vision*, pages 1–18. Springer, 2020.

[55] H. Touvron, T. Lavril, G. Izacard, X. Martinet, M.-A. Lachaux, T. Lacroix, B. Rozière, N. Goyal, E. Hambro, F. Azhar, et al. Llama: Open and efficient foundation language models. *arXiv preprint arXiv:2302.13971*, 2023.

[56] D. Tzionas, L. Ballan, A. Srikantha, P. Aponte, M. Pollefeys, and J. Gall. Capturing hands in action using discriminative salient points and physics simulation. *International Journal of Computer Vision*, 118:172–193, 2016.

[57] J. Wang, F. Mueller, F. Bernard, S. Sorli, O. Sotnychenko, N. Qian, M. A. Otaduy, D. Casas, and C. Theobalt. RGB2Hands: real-time tracking of 3D hand interactions from monocular rgb video. *ACM Transactions on Graphics*, 39(6):1–16, 2020.

[58] K. Wang, Q. Wu, L. Song, Z. Yang, W. Wu, C. Qian, R. He, Y. Qiao, and C. C. Loy. Mead: A large-scale audio-visual dataset for emotional talking-face generation. In *Proceedings of the European Conference on Computer Vision*, 2020.

[59] W. Wang, H.-I. Ho, C. Guo, B. Rong, A. Grigorev, J. Song, J. J. Zarate, and O. Hilliges. 4D-DRESS: A 4D dataset of real-world human clothing with semantic annotations. In *Proceedings of the IEEE/CVF Conference on Computer Vision and Pattern Recognition*, 2024.

[60] S.-E. Wei, J. Saragih, T. Simon, A. W. Harley, S. Lombardi, M. Perdoch, A. Hypes, D. Wang, H. Badino, and Y. Sheikh. VR facial animation via multiview image translation. *ACM Transactions on Graphics*, 38(4):1–16, 2019.

[61] C.-h. Wuu, N. Zheng, S. Ardisson, R. Bali, D. Belko, E. Brockmeyer, L. Evans, T. Godisart, H. Ha, X. Huang, A. Hypes, T. Koska, S. Krenn, S. Lombardi, X. Luo, K. McPhail, L. Millerschoen, M. Perdoch, M. Pitts, A. Richard, J. Saragih, J. Saragih, T. Shiratori, T. Simon, M. Stewart, A. Trimble, X. Weng, D. Whitewolf, C. Wu, S.-I. Yu, and Y. Sheikh. Multiface: A Dataset for Neural Face Rendering. In *arXiv*, 2022. doi: 10.48550/ARXIV.2207.11243. URL https://arxiv.org/abs/2207.11243.

[62] H. Xu, E. G. Bazavan, A. Zanfir, W. T. Freeman, R. Sukthankar, and C. Sminchisescu. GHUM & GHUML: Generative 3D human shape and articulated pose models. In *Proceedings of the IEEE/CVF Conference on Computer Vision and Pattern Recognition*, pages 6184–6193, 2020.

[63] H. Yang, H. Zhu, Y. Wang, M. Huang, Q. Shen, R. Yang, and X. Cao. FaceScape: A large-scale high quality 3D face dataset and detailed riggable 3D face prediction. In *Proceedings of IEEE/CVF Conference on Computer Vision and Pattern Recognition*, June 2020.

[64] J. Yang, J.-S. Franco, F. Hétroy-Wheeler, and S. Wuhrer. Estimation of human body shape in motion with wide clothing. In *Proceedings of the European Conference on Computer Vision*, pages 439–454. Springer, 2016.

[65] T. Yenamandra, A. Tewari, F. Bernard, H. Seidel, M. Elgharib, D. Cremers, and C. Theobalt. i3DMM: Deep implicit 3D morphable model of human heads. In *Proceedings of the IEEE/CVF Conference on Computer Vision and Pattern Recognition*, June 2021.

[66] T. Yu, Z. Zheng, K. Guo, P. Liu, Q. Dai, and Y. Liu. Function4D: Real-time human volumetric capture from very sparse consumer rgbd sensors. In *Proceedings of the IEEE/CVF Conference on Computer Vision and Pattern Recognition*, pages 5746–5756, 2021.

[67] Z. Yu, J. S. Yoon, I. K. Lee, P. Venkatesh, J. Park, J. Yu, and H. S. Park. HUMBI: A large multiview dataset of human body expressions. In *Proceedings of IEEE/CVF Conference on Computer Vision and Pattern Recognition*, June 2020.

[68] S. Yuan, Q. Ye, B. Stenger, S. Jain, and T.-K. Kim. BigHand2.2M benchmark: Hand pose dataset and state of the art analysis. In *Proceedings of the IEEE/CVF Conference on Computer Vision and Pattern Recognition*, pages 4866–4874, 2017.

[69] C. Zhang, S. Pujades, M. J. Black, and G. Pons-Moll. Detailed, accurate, human shape estimation from clothed 3D scan sequences. In *Proceedings of the IEEE/CVF Conference on Computer Vision and Pattern Recognition*, July 2017.

[70] J.-Y. Zhu, T. Park, P. Isola, and A. A. Efros. Unpaired image-to-image translation using cycle-consistent adversarial networks. In *Proceedings of the IEEE/CVF International Conference on Computer Vision*, pages 2223–2232, 2017.

[71] C. Zimmermann, D. Ceylan, J. Yang, B. Russell, M. Argus, and T. Brox. FreiHAND: A dataset for markerless capture of hand pose and shape from single rgb images. In *Proceedings of the IEEE/CVF International Conference on Computer Vision*, pages 813–822, 2019.

[72] X. Zou, X. Han, and W. Wong. CLOTH4D: A dataset for clothed human reconstruction. In *Proceedings of the IEEE/CVF Conference on Computer Vision and Pattern Recognition*, pages 12847–12857, 2023.

[73] M. Zwicker, H. Pfister, J. Van Baar, and M. Gross. EWA splatting. *IEEE Transactions on Visualization and Computer Graphics*, 8(3):223–238, 2002.

